# A Recurrent Neural Network for Word Identification from Continuous Phoneme Strings

**Robert B. Allen**
Bellcore
Morristown, NJ 07962-1910

**Candace A. Kamm**
Bellcore
Morristown, NJ 07962-1910

## Abstract

A neural network architecture was designed for locating word boundaries and identifying words from phoneme sequences. This architecture was tested in three sets of studies. First, a highly redundant corpus with a restricted vocabulary was generated and the network was trained with a limited number of phonemic variations for the words in the corpus. Tests of network performance on a transfer set yielded a very low error rate. In a second study, a network was trained to identify words from expert transcriptions of speech. On a transfer test, error rate for correct simultaneous identification of words and word boundaries was 18%. The third study used the output of a phoneme classifier as the input to the word and word boundary identification network. The error rate on a transfer test set was 49% for this task. Overall, these studies provide a first step at identifying words in connected discourse with a neural network.

## 1 INTRODUCTION

During the past several years, researchers have explored the use of neural networks for classifying spectro-temporal speech patterns into phonemes or other sub-word units (e.g., Harrison & Fallside, 1989; Kamm & Singhal, 1990; Waibel *et al.*, 1989). Less effort has focussed on the use of neural nets for identifying words from the phoneme sequences that these spectrum-to-phoneme classifiers might produce. Several recent papers, however, have combined the output of neural network phoneme recognizers with other techniques, including dynamic time warping (DTW) and hidden Markov models (HMM) (e.g., Miyatake, *et al.*, 1990; Morgan & Bourlard, 1990).

Simple recurrent neural networks (Allen, 1990; Elman, 1990; Jordan, 1986) have been shown to be able to recognize simple sequences of features and have been applied to linguistic tasks such as resolution of pronoun reference (Allen, 1990). We consider whether they can be applied to the recognition of words from phoneme sequences. This paper presents the results of three sets of experiments using recurrent neural networks to locate word boundaries and to identify words from phoneme sequences. The three experiments differ primarily in the degree of similarity between the input phoneme sequences and the input information that would typically be generated by a spectrum-to-phoneme classifier.

## 2 NETWORK ARCHITECTURE

The network architecture is shown in Figure 1. Sentence-length phoneme sequences are stepped past the network one phoneme at a time. The input to the network on a given

time step within a sequence consists of three 46-element vectors (corresponding to 46 phoneme classes) that identify the phoneme and the two subsequent phonemes. The activation of state unit $S_i$ on the step at time $t$ is a weighted sum of the activation of its corresponding hidden unit ($H$) and the state unit's activation on the previous time step, where $\beta$ is the weighting factor for the hidden unit activation and $\mu$ is the state memory weighting factor: $S_{i,t} = \beta H_{i,t-1} + \mu S_{i,t-1}$. In this research $\beta=1.0$ and $\mu=0.5$. The output of the network consists of one unit for each word in the lexicon and an additional unit whose activation indicates the presence of a word boundary.

Weights from the hidden units to the word units were updated based on error observed only at phoneme positions that corresponded to the end of a word (Allen, 1988). The end of the phoneme sequence was padded with codes representing silence. State unit activations were reset to zero at the end of each sentence. The network was trained using a momentum factor of $\alpha=0.9$ and an average learning rate of $\eta=0.05$. The learning rate was adjusted for each output unit proportionally to the relative frequency of occurrence of the word corresponding to that unit.

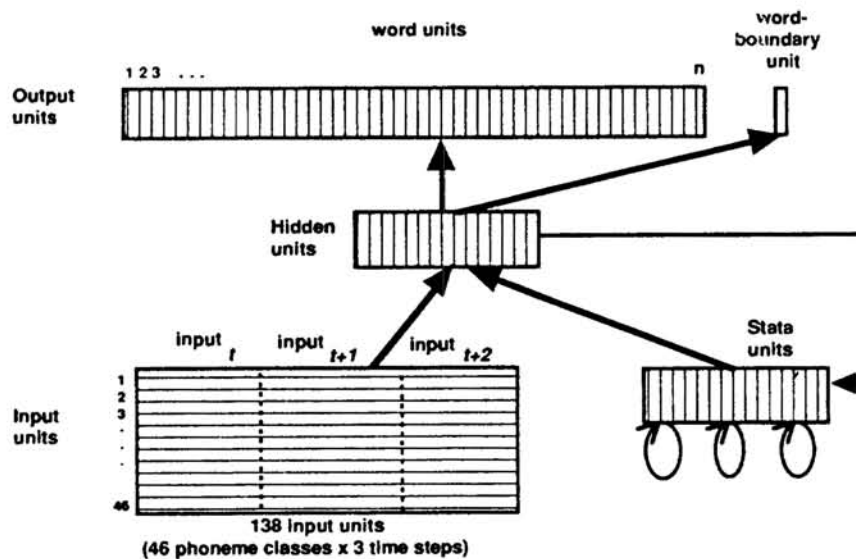

Figure 1:  Recurrent Network for Word Identification

## 3   EXPERIMENT 1: DICTIONARY TRANSCRIPTIONS

### 3.1   PROCEDURE

A corpus was constructed from a vocabulary of 72 words. The words appeared in a variety of training contexts across sentences and the sentences were constrained to a very small set of syntactic constructions. The vocabulary set included a subset of rhyming words. Transcriptions of each word were obtained from Webster's Seventh Collegiate Dictionary and from an American-English orthographic-phonetic dictionary (Shoup, 1973). These transcriptions describe words in isolation only and do not reflect coarticulations that occur when the sentences are spoken. For this vocabulary, 26 of the words had one pronunciation, 19 had two variations, and the remaining 27 had from 3 to 17 variations. The corpus consisted of 18504 sentences of about 7 words each. 6000 of these sentences were randomly selected and reserved as transfer sentences.

The input to the network was a sequence of 46-element vectors designed to emulate the activations that might be obtained from a neural network phoneme classifier with 46 output classes (Kamm and Singhal, 1990). Since a phoneme classifier is likely to generate a set of phoneme candidates for each position in a sequence, we modified the activations in each input vector to mimic this expected situation. Confusion data were obtained from a neural network classifier trained to map spectro-temporal input to an output activation vector for the 46 phoneme classes. In this study, input activations for phonemes that accounted for fewer than 5% of the confusions with the correct phoneme remained set to 0.0, while input activations for phonemes accounting for higher proportions of the total confusions with the correct phoneme were set to twice those proportions, with an upper limit of 1.0. This resulted in relatively high activation levels for one to three elements and activation of 0.0 for the others. Overall, the network had 138 (46×3) input units, 80 hidden units, 80 state units, and 73 output units (one for each word and one boundary detection unit). The network was trained for 50000 sentence sequences chosen randomly (with replacement) from the training corpus. Each sequence was prepared with randomly selected transcriptions from the available dialectal variations.

## 3.2   RESULTS

In all the experiments discussed in this paper, performance of the network was analyzed using a sequential decision strategy. First, word boundaries were hypothesized at all locations where the activation of the boundary unit exceeded a predefined threshold (0.5). Then, the activations of the word units were scanned and the unit with the highest activation was selected as the identified word. By comparing the locations of true word boundaries with hypothesized boundaries, a false alarm rate (i.e., the number of spurious boundaries divided by the number of non-boundaries) was computed. Word error rate was then computed by dividing the number of incorrect words at correctly-detected word boundaries by the total number of words in the transfer set. This word error rate includes both deletions (i.e., missed boundaries) and word substitutions (i.e., incorrectly-identified words at correct boundaries). Total error rate is obtained by summing the word error rate and the false alarm rate.

On the 6000 sentence test set, the network correctly located 99.3% of the boundaries, with a word error rate of 1.7% and a false alarm rate of 0.3% Overall, this yielded a total error rate of 2.0%. To further test the robustness of this procedure to noisy input, three networks were trained with the same procedures as above except that the input phoneme sequences were distorted. In the first network, there was a 30% chance that input phonemes would be duplicated. In a second network, there was a 30% chance that input phonemes would be deleted. In the third network, there was a 70% chance that an input phoneme would be substituted with another closely-related phoneme. Total error rates were 11.7% for the insertion network, 20.9% for the deletion network, and 10.0% for the substitution network.

Even with these fairly severe distortions, the network is moderately successful at the boundary detection/word identification task. Experiment 2 was designed to study network performance on a more diverse and realistic training set.

## 4  EXPERIMENT 2: EXPERT TRANSCRIPTIONS OF SPEECH

### 4.1  PROCEDURE

To provide a richer and more natural sample of transcriptions of continuous speech, training sequences were derived from phonetic transcriptions of 207 sentences from the DARPA Acoustic-Phonetic (AP) speech corpus (TIMIT) (Lamel, *et al.*, 1986). The training set consisted of 4-5 utterances of each of 50 sentences, spoken by different talkers. One other utterance of each sentence (spoken by different talkers) was used for transfer tests. This corpus contained 277 unique words. For training, the transcripts were also segmented into words. When a word boundary was spanned by a single phoneme because of coarticulation (for example, the transcription /haejr/ for the phrase "had your"), the coarticulated phoneme (in this example, /j/) was arbitrarily assigned to only the first word of the phrase. These transcriptions differ from those used in Experiment 1 primarily in the amount of phonemic variation observed at word boundaries, and so provide a more difficult boundary detection task for the network. As in Experiment 1, the input to the network on any time step was a set of three 46-element vectors. The original input vectors (obtained from the phonetic transcriptions) were modified based on the phoneme confusion data described in Section 3.1. The network had 138 (3×46) input units, 80 hidden units, 80 state units, and 278 output units.

### 4.2  RESULTS

After training on 80000 sentence sequences randomly selected from the 207 sentence training set (approximately 320,000 weight updates), the network was tested on the 50 sentence transfer set. With a threshold for the boundary detection unit of 0.5, the network was 87.5% correct at identifying word boundaries and had a false alarm rate of 2.3%. The word error rate was 15.5%. Thus, using the sequential decision strategy, the total error rate was 17.8%.

Considering all word boundaries (i.e., not just the correctly-detected boundaries), word identification was 90.3% correct when the top candidate only (i.e., the output unit with the highest activation) was evaluated and 96.3% for the top three word choices. Because there were instances where boundaries were not detected, but the word unit activations indicated the correct word, a decision strategy that simultaneously considered both the activation of the word boundary unit and the activation of the word units was also explored. However, the distributions of the word unit activations at non-boundary locations (i.e., within words) and at word boundaries overlapped significantly, so this strategy was unsuccessful at improving performance. In retrospect, this result is not very surprising, since the network was not trained for word identification at non-boundaries.

Many of the transfer cases were similar to the training sentences, but some interesting generalizations were observed. For example, the word "beautiful" always appeared in the training set as /bjufəfl/, but its appearance in the test set as /bjufəfUl/ was correctly identified. That is, the variation in the final syllable did not prevent correct identification of the word boundary or the word, despite the fact that the network had seen other instances of the phoneme sequence /Ul/ in the words "woolen" and "football" (second syllable). Of the 135 word transcriptions in the transfer set that were unique (i.e., they did not appear in the training set), the net correctly identified 72% based on the top candidate and 85% within the top 3 candidates. Not surprisingly, performance for the 275 words in the transfer set with non-unique transcriptions was higher, with 96% correct

for the top candidate and 98% for the top 3 choices.

There was evidence that the network occasionally made use of phoneme context beyond word boundaries to distinguish among confusable transcriptions. For example, the words "an", "and", and "in" all appeared in the transfer set on at least one occasion as /ən/, but each was correctly identified. However, many word identification errors were confusions between words with similar final phonemes (e.g., confusing "she" with "be", "pewter" with "order"). This result suggests that, in some instances, the model is not making sufficient use of prior context.

## 5  EXPERIMENT 3: MACHINE-GENERATED TRANSCRIPTIONS

### 5.1  CORPUS AND PROCEDURE

In this experiment, the input to the network was obtained by postprocessing the output of a spectrum-to-phoneme neural network classifier to produce sequences of phoneme candidates. The spectrum-to-phoneme classifier, (Kamm and Singhal, 1990), generates a series of 46-element vectors of output activations corresponding to 46 phoneme classes. The spectro-temporal input speech patterns are stepped past the classifier in 5-ms increments, and the classifier generates output vectors on each step. Since phonemes typically have average durations longer than 5 ms, a postprocessing stage was required to compress the output of the classifier into a sequence of phoneme candidates appropriate for use as input to the boundary detection/word identification neural network.

The training set was a subset of the DARPA A-P corpus consisting of 2 sentences spoken by each of 106 talkers. The postprocessor provided output sequences for the training sentences that were quite noisy, inserting 2233 spurious phonemes and deleting 581 of the 7848 phonemes identified by expert transcription. Furthermore, in 2914 instances, the phoneme candidate with highest average activation was not the correct phoneme. However, this result was not unexpected, since the postprocessing heuristics are still under development. The primary purpose for using the postprocessor output was to provide a difficult input set for the boundary detection/word identification network.

After postprocessing, the highest-activation phoneme candidate sequences were mapped to the input vectors for the boundary detection/word identification network as follows: the vector elements corresponding to the three highest-activation phoneme candidate classes were set to their corresponding average activation values, and the other 43 phoneme classes were set to 0.0 activation. The network had 138 (i.e., 46×3 ) input units, 40 hidden units, 40 state units and 22 output units (21 words and 1 boundary unit). The network was trained for 40000 sentence sequences and then tested on a transfer set consisting of each sentence spoken by a new set of 105 talkers. The sequences in the transfer set were also quite noisy, with 2162 inserted phoneme positions and 775 of 7921 phonemes deleted. Further, the top candidate was not the correct phoneme in 3175 positions.

### 5.2  RESULTS

The boundary detection performance of this network was 56%, much poorer than for the networks with less noisy input. Since the network sometimes identified the word boundary at a slightly different phoneme position than had been initially identified, we implemented a more lenient scoring criterion that scored a "correct" boundary detection

whenever the activation of the boundary unit exceeded the threshold criterion at the true boundary position *or* at the position immediately preceding the true boundary. Even with this looser scoring criterion, only 65% of the boundaries in the transfer set were correctly detected using a boundary detection threshold of 0.5. The false alarm rate was 9% and the word error rates were 40%, yielding a total error rate of 49%. This is much larger than the error rate for the network in Experiment 2. This difference may be explained in part by the presence of insertions in the input stream in this experiment as compared to Experiment 2, which had no insertions. The results of Experiment 2 indicated that this recurrent architecture has a limited capacity for considering past information (i.e., as evidenced by substitution errors such as "she" for "be" and "pewter" for "order"). As a result, poorer performance might be expected when the information required for word boundary detection or word identification spans longer input sequences, as occurs when the input contains extra vectors representing inserted phonemes.

## 5.3    NON-RECURRENT NETWORK

To evaluate the utility of the recurrent network architecture for this task, a simple non-recurrent network was trained using the same training set. In addition to the *t, t+1* and *t+2* input slots (Fig. 1), the non-recurrent network also included *t-1* and *t-2* slots, in an attempt to match some of the information about past context that may be available to the recurrent network through the state unit activations. Thus, the input required 230 (i.e., 5×46) input units. The network had no state units, 40 hidden units and 22 output units, and was trained through 40000 sentence sequences. On the transfer set, using a boundary detection threshold of 0.5, 75% of the word boundaries were correctly detected, and a false alarm rate of 31%. The word error rate was 60%. Thus, the recurrent net performed consistently better than this non-recurrent network both in terms of fewer false alarms and fewer word errors, despite the fact that the non-recurrent network had more weights. These results suggest that recurrence is important for the boundary and word identification task with this noisy input set.

## 6    DISCUSSION AND FUTURE DIRECTIONS

The current results suggest that this neural network model may provide a way of integrating lower-level spectrum-to-phoneme classification networks and phoneme-to-word classification networks for automatic speech recognition. The results of these initial experiments are moderately encouraging, demonstrating that this architecture can be used successfully for boundary detection with moderately large (200-word) and noisy corpora, although performance drops significantly when the input stream has many inserted and deleted phonemes. Furthermore, these experiments demonstrate the importance of recurrence.

Many unresolved questions about the application of this model for the word boundary/word identification task remain. The performance of this model needs to be compared with that of other techniques for locating and identifying words from phoneme sequences (for example, the two-level dynamic programming algorithm described by Levinson *et al.*, 1990).

Word-identification performance of the model (based on the output class with highest activation) is far from perfect, suggesting that additional strategies are needed to improve performance. First, word identification errors substituting "she" for "be" and "pewter" for "order" suggest that the network sometimes uses information only from one or two previous time steps to make word choices. Efforts to extend the persistence of

information in the state units beyond this limit may improve performance, and may be especially helpful when the input is corrupted by phoneme insertions. Another possible strategy for improving performance would be to use the locations and identities of words whose boundaries and identities can be hypothesized with high certainty as "islands of reliability". These anchor points could then help determine whether the word choice at a less certain boundary location is a reasonable one, based on features like word length (in phonemes) or semantic or syntactic constraints. In addition, an algorithm that considers more than just the top candidate at each hypothesized word position and that uses semantic and syntactic constraints for reducing ambiguity might be prove more robust than the single-best-choice word identification strategy used in the current experiments. Schemes that attempt to identify word sequences without specifically locating word boundaries should be explored. The question of whether this network architecture will scale to successfully handle still larger corpora and realistic applications also requires further study. These unresolved issues notwithstanding, the current work demonstrates the feasibility of an integrated neural-based system for performing several levels of processing of speech, from spectro-temporal pattern classification to word identification.

## References

Allen, R.B. Sequential connectionist networks for answering simple questions about a microworld. *Proceedings of the Cognitive Science Society*, 489-495, 1988.

Allen, R.B. Connectionist language users. *Connection Science*, **2**, 279-311, 1990.

Elman, J. L. Finding structure in time. *Cognitive Science*, **14**, 179-211, 1990.

Harrison, T. and Fallside, F. A connectionist model for phoneme recognition in continuous speech. *Proc. ICASSP 89*, 417-420, 1989.

Jordan, M. I. Serial order: A parallel distributed processing approach. *(Tech. Rep. No. 8604)*. San Diego: University of California, Institute for Cognitive Science, 1986.

Kamm, C. and Singhal, S. Effect of neural network input span on phoneme classification, *Proc. IJCNN June 1990*, **1**, 195-200, 1990.

Lamel, L., Kassel, R. and Seneff, S. Speech database development: Design and analysis of the acoustic-phonetic corpus. *Proc. DARPA Speech Recognition Workshop*, 100-109, 1986.

Levinson, S. E., Ljolje, A. and Miller, L. G. Continuous speech recognition from a phonetic transcription. *Proc. ICASSP-90*, 93-96, 1990.

Miyatake, M., Sawai, H., Minami, Y. and Shikano, H. Integrated training for spotting Japanese phonemes using large phonemic time-delay neural networks. *Proc. ICASSP 90*, 449-452, 1990.

Morgan, N. and Bourlard, H. Continuous speech recognition using multilayer perceptrons with hidden Markov models. *Proc. ICASSP 90*, 413-416, 1990.

Shoup, J. E. American English Orthographic-Phonemic Dictionary. NTIS Report AD763784, 1973.

Waibel, A., Hanazawa, T., Hinton, G., Shikano, K. and Lang, K. Phoneme recognition using time-delay neural networks. *IEEE Trans. ASSP*, **37**, 328-339, 1989.

Webster's Seventh Collegiate Dictionary. Springfield, MA: Merriam Company, 1972.
